# Deep Spatio-Temporal Architectures and Learning for Protein Structure Prediction

**Pietro Di Lena, Ken Nagata, Pierre Baldi**
Department of Computer Science, Institute for Genomics and Bioinformatics
University of California, Irvine
{pdilena,knagata,pfbaldi}@[ics.]uci.edu

## Abstract

Residue-residue contact prediction is a fundamental problem in protein structure prediction. Hower, despite considerable research efforts, contact prediction methods are still largely unreliable. Here we introduce a novel deep machine-learning architecture which consists of a multidimensional stack of learning modules. For contact prediction, the idea is implemented as a three-dimensional stack of Neural Networks $\text{NN}_{ij}^k$, where $i$ and $j$ index the spatial coordinates of the contact map and $k$ indexes "time". The temporal dimension is introduced to capture the fact that protein folding is not an instantaneous process, but rather a progressive refinement. Networks at level $k$ in the stack can be trained in supervised fashion to refine the predictions produced by the previous level, hence addressing the problem of vanishing gradients, typical of deep architectures. Increased accuracy and generalization capabilities of this approach are established by rigorous comparison with other classical machine learning approaches for contact prediction. The deep approach leads to an accuracy for difficult long-range contacts of about 30%, roughly 10% above the state-of-the-art. Many variations in the architectures and the training algorithms are possible, leaving room for further improvements. Furthermore, the approach is applicable to other problems with strong underlying spatial and temporal components.

## 1 Introduction

*Protein structure prediction* from amino acidic sequence is one of the grand challenges in Bioinformatics and Computational Biology. To date, the more accurate and reliable computational methods for protein structure prediction are based on *homology modeling* [27]. Homology-based methods use similarity to model the unknown target structure using known template structures. However, when good templates do not exist in protein structure repositories or when sequence similarity is very poor–which is often the case–homology modeling is no more effective. This is the realm of *ab initio modeling* methods, which attempt to recover three-dimensional protein models more or less from scratch. Because the structure of proteins is invariant under translations and rotations, it is useful to consider structural representations that do not depend on cartesian coordinates. One such representation is the *contact map*, essentially a sparse binary matrix representing which amino acids are in contact in the 3D structure. While contact map prediction can be viewed as a sub-problem in protein structure prediction, it is well known that it is essentially equivalent to protein structure predictions since 3D structures can be completely recovered from sufficiently large subsets of true contacts [20, 26, 23]. Furthermore, even small sets of correctly predicted contacts can be useful for improving ab initio methods [25]. In short, contact map prediction plays a fundamental role in protein structure prediction and most of the state-of-the art contact predictors use some form of machine learning. Contact prediction is assessed every two years in the CASP experiments [9, 15]. However, despite considerable efforts, the accuracy of the best predictors at CASP rarely exceeds

20% for long-range contacts, suggesting major room for improvements. Simulations suggest that this accuracy ought to be increased to about 35% in order to be able to recover good 3D structures.

There are two main issues arising in contact prediction that have not been addressed systematically: (1) Residue contacts are not randomly distributed in native protein structures, rather they are spatially correlated. Current contact predictors generally do not take into account these correlations, not even at the local level, since the contact probability for a residue pair is typically learned/inferred independently of the contact probabilities in the neighborhood of the pair. (2) Proteins do not assume a 3D conformation instantaneously, but rather through a dynamic folding process that progressively refines the structure. In contrast, current machine learning approaches attempt to learn contact map probabilities in a single step. To address these issues, here we introduce a new machine-learning deep architecture, designed as a deep stack of neural networks, in such a way that each level in the stack receives in input and refines the predictions produced at the previous level. Each level can be trained in a fully supervised fashion on the same set of target contacts/non-contacts, thus overcoming the gradient vanishing problem, typical of deep architectures. The idea of layering learning modules, such that the outputs of previous layers are fed in input to the next layers, is not completely new and it has been applied in different contexts, particularly to computer vision detection problems [4, 10, 12, 22]. However the techniques developed in visual detection cannot be directly applied to contact prediction due to the intrinsic difference of such problems: protein sequences have different lengths, thus it is not possible to process the entire sequence at once in the network input, as it is done for images. The present work represents, to our knowledge, the first attempt to introduce spatial correlation in protein contact prediction.

## 2  Data preparation

### 2.1  Contact definition and evaluation criteria

We define two residues to be in contact if the Euclidean distance between their $C_\beta$ atoms ($C_\alpha$ for Glycines) is lower than 8Å. This is the contact definition adopted for the contact prediction assessment in CASP experiments [15]. The protein *map of contact* (or contact map) provides a two-dimensional translation and rotation invariant representation of the protein three-dimensional structure. The information content of the contact map is not uniform within different regions of the map. Three distinct classes of contacts can be defined, depending on the linear sequence separation between the residues: (1) long-range contacts, with separation $\geq 24$ residues; (2) medium-range contacts, with separation between 12 and 23 residues; and (3) short-range contacts, with separation between 6 and 11 residues. Contacts between residues separated by less than 6 residues are dense and can be easily predicted from the secondary structure. Conversely, the sparse long-range contacts are the most informative and also the most difficult to predict. Thus, as in the CASP experiments, we focus primarily on long-range contact prediction for performance assessment. The contact prediction performance is evaluated using the standard *accuracy* measure [15]: $Acc = \text{TP}/(\text{TP}+\text{FP})$, where TP and FP are the true positive and false positive predicted contacts, respectively. The $Acc$ measure is computed for the sets of $L/5$, $L/10$ and 5 top scored predicted pairs, where $L$ is the length of the domain sequence. The most widely accepted measure of performance for contact prediction assessment is $Acc$ for $L/5$ pairs and sequence separation $\geq 24$ [15].

### 2.2  Training and test sets

In order to asses the performance of our method, a training and a test set of protein domains are derived from the ASTRAL database [6]. We extract from the ASTRAL release 1.73 the precompiled set of protein domains with less than 20% pairwise sequence identity. We select only the domains belonging to the main SCOP [17] classes (All-Alpha, All-Beta, Alpha/Beta and Alpha+Beta). We exclude domains of length less than 50 residues, domains with multiple 3D structures, as well as non-contiguous domains (including those with missing backbone atoms). We further filter this list by selecting just one representative domain–the shortest one–per SCOP family. This yields a training set of 2,191 structures (the list of protein domains can be found as supplementary material of [8]). For performance assessment purposes, this set is partitioned into 10 disjoint groups of roughly the same size and average domain lengths, so that no domains from two distinct groups belong to the same SCOP fold. As a result, the 10 sets do not share any structural or sequence similarity, providing

a high-quality benchmark for ab initio prediction. Model performance is assessed using a standard 10-fold cross-validation procedure. In all our tests, the accuracy results on training/test are averaged over the 10 cross-validation experiments.

## 2.3 Feature and training example selection

In this work, we do not attempt to determine the best static input features for contact prediction. Rather, we focus on a minimal set of features commonly used in machine learning-based contact prediction [11, 2, 21, 7, 24, 5]. Each residue in the protein sequence is described by a feature vector encoding three sources of information (for a total of 25 values): *evolutionary information* in the form of profiles (20 values, one for each amino acid type), *predicted secondary structure* (3 binary values, $\beta$-sheet or $\alpha$-helix or coil), and *predicted solvent accessibility* (2 binary values, buried or exposed). The profiles are computed using PSI-BLAST [1] with an E-value cutoff equal to 0.001 and up to ten iterations against the non redundant protein sequence database (NR). The secondary structure is predicted with SSPRO [18] and the solvent accessibility with ACCPRO [19]. For a pair of residues, these features are included in the network input by using a 9-residue long sliding window centered at each residue in the pair. In our Deep NN, these features represent the spatial features (Section 3).

The uneven distribution of positive (residue pairs in contact) and negative (residue pairs not in contact) examples in native protein structures requires some rebalancing of the training data. For each training structure we randomly select 20% of the negative examples, while keeping all the positive examples. We do not include in our set of selected examples residue pairs with sequence separation less than 6. All the methods compared in Section 4 are trained on exactly the same sets of examples.

# 3 Deep Spatio-Temporal Neural Network (DST-NN) architecture

In the specific implementation used in the simulations, the DST-NN architecture consists of a three-dimensional stack of neural networks $\text{NN}_{ij}^k$, where $i$ and $j$ are the usual spatial coordinates of the contact map, and $k$ is a "temporal" index. All the neural networks in the stack have the same topology (same input, hidden, and output layer sizes) with a single hidden layer, and a single sigmoidal output unit estimating the probability of contact between $i$ and $j$ at the level $k$ (Figure 1(a) and 1(b)). Furthermore, in this implementation, all the networks in the level $k$ have the same weights (weight sharing). Each level $k$ can be trained in a fully supervised fashion, using the same contact maps as targets. In this way, each level of the deep architecture represents a distinct contact predictor. The inputs into $\text{NN}_{ij}^k$ can be separated into purely spatial inputs, and temporal inputs (which are not purely temporal but include also a spatial component). For fixed $i$ and $j$, the purely spatial inputs are identical for all levels $k$ in the stack, hence they do not depend on "time". These purely spatial inputs include evolutionary profiles, predicted secondary structure, and solvent accessibility in a window around residue $i$ and residue $j$. These are the standard inputs used by most other predictors which attempt to predict contacts in one shot and are described in more detail in Section 2.3. The temporal inputs, on the other hand, are novel.

## 3.1 Temporal Features

The temporal inputs for $\text{NN}_{ij}^k$ correspond to the outputs of the networks $\text{NN}_{rs}^{k-1}$ at the previous level in the stack, where $r$ and $s$ range over a neighborhood of $i$ and $j$. Here we use a neighborhood of radius 4 centered at $(i, j)$. The temporal features capture the idea that residue contacts are not randomly distributed in native protein structures, rather they are spatially correlated: a contacting residue pair is very likely to be in the proximity of a different pair of contacting residues. For instance, a comparison of the contact proximity distribution (data not shown) for long-range residue pairs in contact and not in contact shows that over 98% of the contacting residue pairs are in the proximity of at least one additional contact, compared to 30% for non-contacting residue pairs, within a neighborhood of radius 4. Although the contact predictions at a given level of the stack are inaccurate, the contact probabilities included in the temporal feature vector can still provide some rough estimation of the contact distribution in a given neighborhood.

Thus, in short, while our model is not necessarily meant to simulate the physical folding process, the stack is used to organize the prediction in such a way that each level in the stack is meant to refine the predictions produced by the previous levels, integrating information over both space and

time. In particular, through the temporal inputs the architecture ought to be able to capture spatial correlations between contacts, at least over some range.

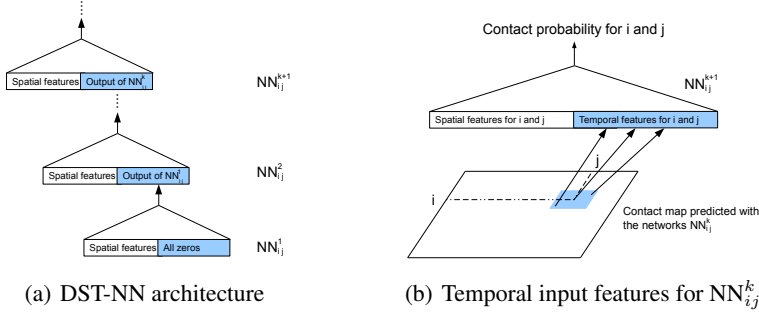

(a) DST-NN architecture         (b) Temporal input features for $NN^k_{ij}$

Figure 1: DST-NN architecture. (a) Overview. Each $NN^k_{ij}$ represents a feed-forward neural network trainable by back-propagation. (b) For a pair of residues $(i, j)$, the temporal inputs into $NN^{k+1}_{ij}$ consist of the contact probabilities produced by the network at the previous level over a neighborhood of $(i, j)$.

## 3.2 Deep Learning

Training deep multi-layered neural networks is generally hard, since the error gradient tends to vanish or explode with a high number of layers [16]. In contrast, in the proposed model, the learning capabilities are not directly degraded by the depth of the stack, since each level of the stack can be trained in a supervised fashion using true contact maps to provide the targets. In this way, training can be performed incrementally, by adding a new layer to the stack. More precisely, the weights of the first level network, $NN^1_{ij}$, are randomly initialized and the temporal feature vector is set to 0. The first network $NN^1_{ij}$ is then trained for one epoch on the given set of examples. The weights of $NN^1_{ij}$ are then used to initialize the weights of $NN^2_{ij}$ and the predictions obtained with $NN^1_{ij}$ are used to setup the temporal feature vector of $NN^2_{ij}$. The network $NN^2_{ij}$ is then trained for one epoch on the same set of examples used for $NN^1_{ij}$ and this procedure is repeated up to a certain depth. We have experimented with several variations of this training procedure, such as randomization of the weights for each new network in the stack, training each network in the stack for more than one epoch, growing the stack up to a maximum number of training epochs (one network for each epoch), or growing it to a smaller depth but then repeating the training procedure through one or more epochs. In Section 4.2 we discuss and compare such different training strategies. In Section 5 we discuss some possible variants and generalizations of the full architecture. In any case, this approach enables training very deep networks (e.g. with maximal values of $k$ up to 100, corresponding to a global neural network architecture with 300 layers).

# 4 Results

## 4.1 Performance comparison

Here we investigate the learning and generalization capabilities of the DST-NN model, and compare it with plain three-layer Neural Network (NN) models, as well as 2D Recurrent Neural Network (RNN) models, which are two of the most widely used machine learning approaches for contact prediction [11, 2, 21, 24]. Here, the NN model is perfectly equivalent to the NNs implemented in the DST-NN architecture, except for the temporal feature vector (which is missing in the NN implementation). All three methods are trained with a standard on-line back-propagation procedure using exactly the same set of examples and the same input features (Section 2.3).

One of the most typical problem in neural network design is related to the issue of choosing, for a given classification problem, the most appropriate network size (i.e. typically the hidden layer size, which affects the total number of connections in the network). The learning time and the

generalization capabilities of the particular neural network model are highly affected by the network size parameter. In order to take into account the intrinsic incomparable capabilities of the different DST-NN, NN, and RNN architectures, we perform our tests by considering a range of exponentially increasing hidden layer sizes (4,8,16,32,64, and 128 units) for each architecture. The total number of connection weights for each architecture in function of the hidden layer size, as well as the time needed to perform one training epoch, are shown in Table 1.

Figure 2 shows the learning curves of the three methods as a function of the training epoch and the different hidden layer sizes. We show the cross-training average accuracy on both training sets (continuous line) and test sets (dotted line). The learning curves in Figure 2 show the generalization performance with respect to the contact prediction accuracy on L/5 long range contacts; the accuracy of prediction on long range contacts is the most widely accepted evaluation measure for contact prediction and it provides a better estimate of the prediction performance than the training/testing error. Since very large training epochs are infeasible in terms of time for the RNN model (see Table 1), for the aim of comparison, we trained each method for a maximum of 100 epochs. In Table 2 we summarize the prediction performance of the three machine learning methods by showing the maximum average accuracy achieved in testing over 100 training epochs.

From Figure 2, the DST-NN has overall higher storage and generalization capacity than NN and RNN. In particular, for hidden layer sizes larger than or equal to 8, the DST-NN performance are superior to those of NN and RNN, regardless of their sizes. Moreover, note that hidden layer sizes larger than 32 do not increase the generalization capabilities of any one of the three methods (Table 2). The counterintuitive learning curves of the RNN for hidden layer sizes larger than 8 can be explained by considering the structure of the RNN architecture. The RNN model exploits a recursive architecture that suffers, as general deep architectures, from the problem of gradient vanishing/explosion. In order to overcome this problem the authors of [2] use a modified form of gradient descent, by which the delta-error for back-propagation is mapped into a piecewise linear interval; this prevents the delta-error from becoming too small or too large. The boundaries of the interval have been tuned for very small hidden layers (private communication). In our experiments, we use the same boundaries for all the tested hidden layer sizes and, apparently, these proved to be ineffective for hidden layer sizes larger than or equal to 16. In comparison, we remark again that the DST-NN is unaffected by the gradient vanishing problem, even for very deep stacks. From Figure 2, we notice that the DST-NN tends to overfit the training data more easily than the NN. For instance, we notice some small overfitting for the DST-NN starting with hidden layer size 32, while the NN starts to show some small overfitting only at hidden layer size 128. On the contrary, the RNN does not show any sign of overfitting in 100 epochs of training, regardless the hidden layer size in the tested range, and the performance in training is somewhat equivalent to the performance in testing. As a final consideration, from Table 2, the NN and RNN best performance on L/5 long range contacts reflect quite well the state-of-the-art in contact prediction [9, 15] with an accuracy in the 21-23% range. In contrast, the DST-NN architecture achieves a maximum accuracy of %29 which represents a significant improvement over the state-of-the-art. As a visual example, Figure 3 shows the best predictions obtained by each method on a target domain in our data set. Despite the three methods achieve exactly the same accuracy (0.6) on the top-scored L/5 long range contacts, it is evident that the DST-NN provides an overall better prediction of the contact map topology.

Table 1: Connection weights and training times

| HL size | DST-NN | | NN | | RNN | |
|---|---|---|---|---|---|---|
| | #Conn | Time | #Conn | Time | #Conn | Time |
| 4 | 2,133 | ∼6m | 1,809 | ∼1m | 17,169 | ∼1h30min |
| 8 | 4,265 | ∼10m | 3,617 | ∼3m | 19,105 | ∼2h |
| 16 | 8,529 | ∼15m | 7,233 | ∼5m | 22,977 | ∼2h40m |
| 32 | 17,057 | ∼26m | 14,465 | ∼8m | 30,721 | ∼3h20m |
| 64 | 34,113 | ∼1h20m | 28,929 | ∼15m | 46,209 | ∼4h50m |
| 128 | 68,225 | ∼2h | 57,857 | ∼28m | 77,185 | ∼7h |

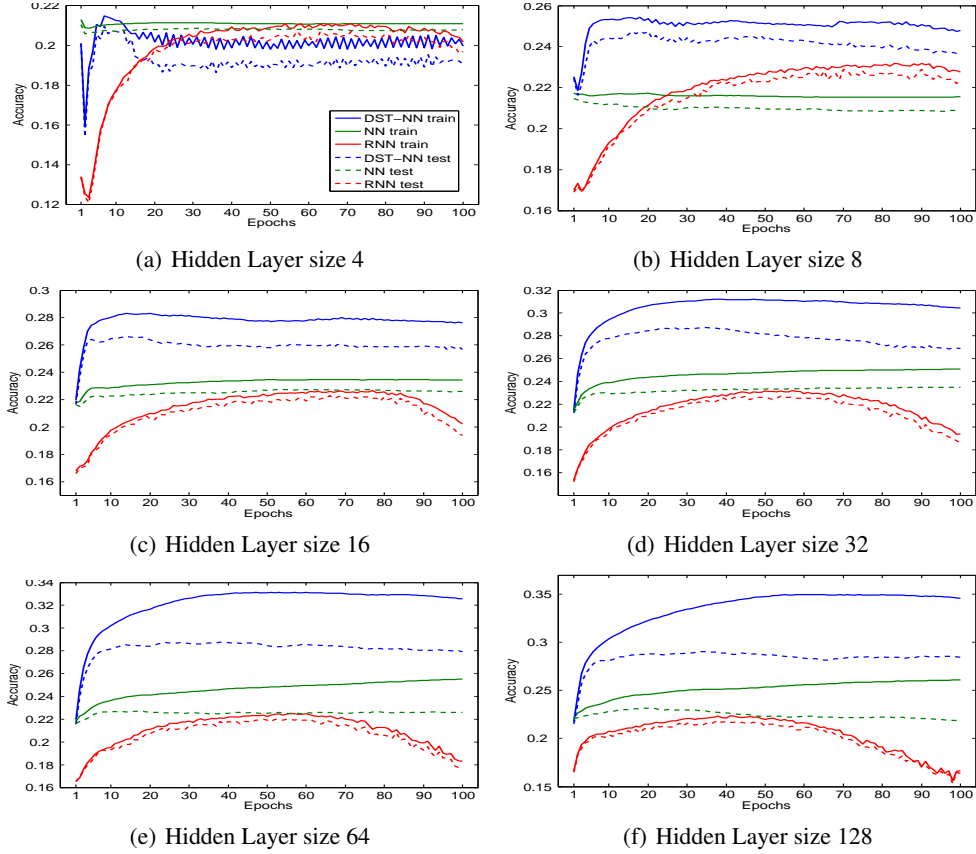

Figure 2: Learning curves of different machine learning methods

Table 2: Best prediction performance

| HL size | DST-NN | | | NN | | | RNN | | |
|---|---|---|---|---|---|---|---|---|---|
| | L/5 | L/10 | Best5 | L/5 | L/10 | Best5 | L/5 | L/10 | Best5 |
| 4 | 0.21 | 0.23 | 0.26 | 0.21 | 0.24 | 0.27 | 0.21 | 0.23 | 0.25 |
| 8 | 0.25 | 0.27 | 0.29 | 0.21 | 0.24 | 0.27 | 0.23 | 0.26 | 0.29 |
| 16 | 0.27 | 0.30 | 0.33 | 0.23 | 0.26 | 0.28 | 0.22 | 0.25 | 0.29 |
| 32 | 0.29 | 0.32 | 0.35 | 0.23 | 0.26 | 0.29 | 0.23 | 0.26 | 0.29 |
| 64 | 0.29 | 0.33 | 0.37 | 0.23 | 0.25 | 0.28 | 0.22 | 0.25 | 0.28 |
| 128 | 0.29 | 0.33 | 0.36 | 0.23 | 0.25 | 0.28 | 0.22 | 0.25 | 0.28 |

## 4.2 Training strategies comparison

Here we compare the generalization performance of the DST-NN under different training strategies. Since the training time for the DST-NN increases substantially with the size of the hidden layers, in these tests we consider only hidden layers of size 16 and 32. On the other end, as shown in Table 2, a hidden layer of size 32 does not limit the generalization performance of our method in comparison to larger sizes. As in the previous section, we show the performance of the different training strategies in terms of learning curves (Figure 4) and maximum achievable accuracy in testing (Table 3).

Recall that, according to our general training strategy, when a new network is added to the stack its initial connection weights are copied from the previous-level network in the stack. Moreover, each network is trained on exactly the same set of examples. Thus, a natural question is to which extent the randomization, in terms of both connection weights and training examples, affects the network learning capabilities. As shown in Figure 4(a)(b), under weight randomization (DST-NN$_1$), the DST-NN gets stuck in local minima and the best prediction performance are comparable to those of NN

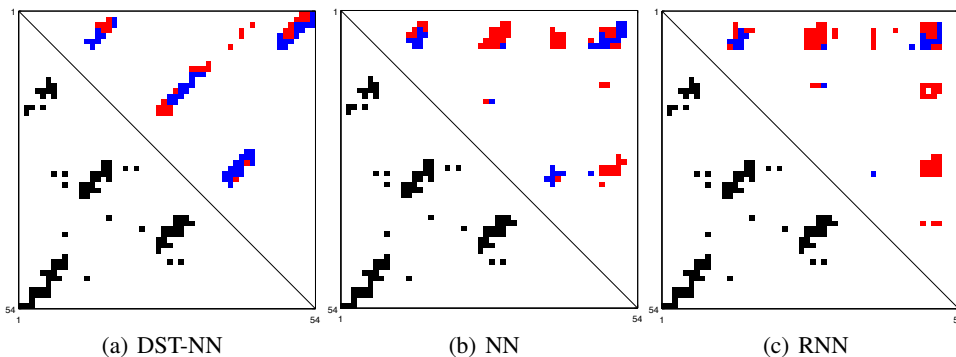

(a) DST-NN&emsp;&emsp;&emsp;&emsp;(b) NN&emsp;&emsp;&emsp;&emsp;(c) RNN

Figure 3: Predicted contacts at sequence separation $\geq 6$ for the d1igqa_ domain. In all three figures, the lower triangle shows the native contacts (black dots). The blue and red dots in the upper triangle represent the correctly (blue) and incorrectly (red) predicted contacts among the N top-scored residue pairs, where N is the number of native contacts at sequence separation $\geq 6$. All three methods achieve 0.6 accuracy on the top L/5 long range contacts.

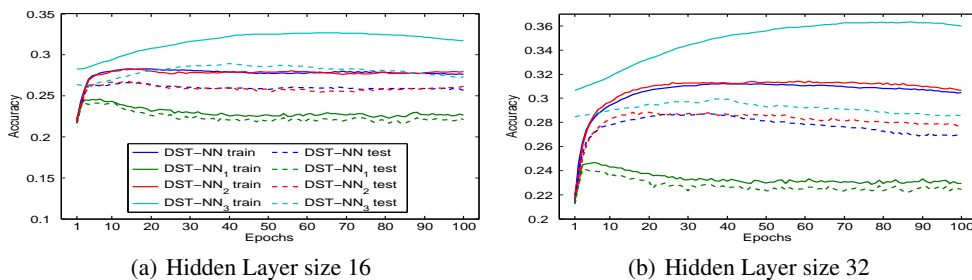

(a) Hidden Layer size 16&emsp;&emsp;&emsp;&emsp;(b) Hidden Layer size 32

Figure 4: Learning curves of different training strategies

and RNN (Table 2 and Table 3). On the other hand, under weight randomization, the DST-NN does not show any sign of overfitting and the training performance is similar to the testing performance, as for the RNN in the previous section. Conversely, randomized selection of the training examples (DST-NN$_2$) does not affect the performance of the DST-NN. However, this training strategy seems to be slightly less stable than our general strategy, since the standard deviation of the accuracy over the ten training/testing sets is slightly higher (data not shown). In these tests, according to our general training strategy, each network in the stack has been trained for one single epoch. The approach of training each network for more than one single epoch leads to slightly better accuracy ($< 1\%$ of improvement) at the cost of a larger training time (data not shown).

Another natural issue concerning DST-NNs is whether the depth of the stack affects the generalization capabilities of the model. To assess this issue, we train a new DST-NN by limiting the depth of the stack to a fixed number of networks and then repeating the training procedure up to 100 epochs (DST-NN$_3$). For this test, we use a limit size of 20 networks, which roughly corresponds to the interval with highest learning peaks for hidden layer size 16 (see Figure 2). Due to the increased training time for this model (20 times slower), testing different stack depths is not practical. For this training strategy, the randomization of the weights for each newly added network in the stack does not produce any dramatic loss in prediction accuracy, although the performance results are slightly lower than those obtained by using our general weight initialization strategy (data not shown). As shown in Figure 4 and Table 3, although more time consuming, this training technique allows an improvement of approximatively 2% points of accuracy with respect to our general training approach (at least for a hidden layer of size 16). For this reason, restarting the training on a fixed size stack is more advantageous in terms of prediction performance than having a very deep stack. Unfortunately, the optimal stack depth is very likely related to the specific classification problem and it cannot be inferred a priori from the architecture topology.

Table 3: Best prediction performance

| Method | HL 16 | | | HL 32 | | |
|---|---|---|---|---|---|---|
| | L/5 | L/10 | Best5 | L/5 | L/10 | Best5 |
| DST-NN | 0.27 | 0.30 | 0.33 | 0.29 | 0.32 | 0.35 |
| DST-NN$_1$ | 0.24 | 0.27 | 0.30 | 0.24 | 0.27 | 0.29 |
| DST-NN$_2$ | 0.27 | 0.30 | 0.33 | 0.29 | 0.33 | 0.36 |
| DST-NN$_3$ | 0.29 | 0.32 | 0.35 | 0.30 | 0.33 | 0.37 |

## 5   Concluding remarks

We have presented a novel and general deep machine-learning architecture for contact prediction, implemented as a stack of Neural Networks $NN_{ij}^k$ with two spatial dimensions and one temporal dimension. The stack architecture is used to organize the prediction in such a way that each level in the stack can receive in input, through the temporal feature vectors, and refine the predictions produced by the previous stages in the stack. This approach is closer to the characteristics of the folding process, where the folded state is dynamically attained through a series of local refinements. While our architecture is not meant to simulate the folding process, the idea to model the contact prediction in a multi-level fashion seems more natural than the traditional single-shot approach. This is confirmed by the improved generalization capabilities and accuracy of the DST-NN model, which have been demonstrated by rigorous comparison against other approaches.

The proposed architecture is somewhat general and it can be adopted as a starting point for more sophisticate methods for contact prediction or other problems. For instance, while the elementary learning modules of the architecture are implemented using neural networks, it is clear that these could be replaced by other models, such as SVMs. Moreover, here we considered a simple square neighborhood for encoding the contact predictions in the temporal feature vector; more complex relationships could be discovered by exploiting different topologies for such feature vector . For example, different secondary structure elements tend to form specific contacting patterns and such patterns could be directly implemented in one or more specific feature vectors (see, for example, [8]). Another property of our DST-NN approach is that each level can be trained in supervised fashion. While we have used the true contact map as the target for all the levels in the architecture, it is clear that different targets could be used at different levels [3]. For instance, experimental or simulation data[1] on protein folding could be used to generate contact maps at different stages of folding and use those as targets. Different variations based on these ideas are currently under investigation.

The DST-NN approach is in fact a special case of the DAG-RNN approach described in [2] and relies on an underlying directed acyclic graph (DAG) to organize the computations. For these reasons, one could also imagine architectures based on a higher-dimensional stack of learning modules, for instance a stack of the form $NN_{ijk}^{lm}$ where the spatial coordinates are three-dimensional, and the "temporal" coordinates are two-dimensional with a connectivity that ensures the absence of directed cycles (the temporal connections running only from the "past" towards the "future"). DST-NNs of the form $NN_i^k$, with one spatial and one temporal coordinate, could be applied to sequence problems, for instance to the prediction of secondary structure or relative solvent accessibility. Likewise, DST-NNs of the form $NN_{ijk}^l$, with three spatial and one temporal coordinate, could be applied, for instance, to problems in weather forecasting [13] or trajectory prediction in robot movements [14].

## References

[1] Altschul,S.F., Madden,T.L., Schäffer,A.A., Zhang,J., Zhang,Z., Miller,W., Lipman, D.J. (1997) Gapped BLAST and PSI-BLAST: a new generation of protein database search programs, *Nucleic Acids Res.*, **25**(17), 3389-3402.

[2] Baldi,P., Pollastri,G. (2003) The Principled Design of Large-Scale Recursive Neural Network Architectures-DAG-RNNs and the Protein Structure Prediction Problem, *Journal of Machine Learning Research*, **4**, 575-602.

---

[1]http:www.dynameomics.org

[3] Baldi,P. (2012) Boolean Autoencoders and Hypercube Clustering Complexity, *Designs, Codes, and Cryptography*, **65**, 383-403.

[4] Bengio,Y., Lamblin,P., Popovici,D., Larochelle,H. (2006) Greedy Layer-Wise Training of Deep Networks. Proceedings of the 20th Annual Conference on Neural Information Processing Systems (NIPS 2006), 153-160.

[5] Björkholm,P., Daniluk,P., Kryshtafovych,A., Fidelis,K., Andersson,R., Hvidsten,T.R. (2009) Using multi-data hidden Markov models trained on local neighborhoods of protein structure to predict residue-residue contacts. *Bioinformatics*, **25**, 1264-1270.

[6] Chandonia,J.M., Hon,G., Walker,N.S., Lo Conte,L., Koehl,P., Levitt, M., Brenner, S.E. (2004) The AS-TRAL Compendium in 2004, *Nucl. Acids Res.* , **32**(suppl 1), D189-D192.

[7] Cheng,J., Baldi,P. (2007) Improved residue contact prediction using support vector machines and a large feature set, *BMC Bioinformatics*, **8**, 113.

[8] Di Lena,P., Nagata,K., Baldi,P. (2012) Deep Architectures for Protein Contact Map Prediction, *Bioinformatics*, **28**, 2449-2457.

[9] Ezkurdia,I., Graña,O., Izarzugaza,J.M., Tress,M.L. (2009) Assessment of domain boundary predictions and the prediction of intramolecular contacts in CASP8, *Proteins*, **77**(suppl 9), 196-209

[10] Farabet,C. Couprie,C., Najman,L., LeCun,Y. (2012) Scene Parsing with Multiscale Feature Learning, Purity Trees, and Optimal Covers. Proceedings of the 29th International Conference on Machine Learning (ICML 2012).

[11] Fariselli,P.,Olmea,O.,Valencia,A.,Casadio,R. (2001) Progress in predicting inter-residue contacts of proteins with neural networks and correlated mutations. *Proteins* **5**, 157-162.

[12] Heitz,G., Gould,S., Saxena,A., Koller,D. (2008) Cascaded Classification Models: Combining Models for Holistic Scene Understanding. Proceedings of the 22nd Annual Conference on Neural Information Processing Systems (NIPS 2008), 641-648.

[13] Hsieh,W. (2009) Machine Learning Methods in the Environmental Sciences: Neural Networks and Kernels. Cambridge University Press, NY, USA.

[14] Jetchev,N., Toussaint,M. (2009) Trajectory prediction: learning to map situations to robot trajectories. Proceedings of the 26th Annual International Conference on Machine Learning, 449-456.

[15] Kryshtafovych,A., Fidelis,K., Moult,J. (2011) CASP9 results compared to those of previous CASP experiments, *Proteins*, In press.

[16] Larochelle,H., Bengio,J., Louradour,J., Lamblin,P. (2009) Exploring Strategies for Training Deep Neural Networks *Journal of Machine Learning Research*, **10**, 1-40.

[17] Murzin,A.G., Brenner,S.E., Hubbard,T., Chothia,C. (1995) SCOP: a structural classification of proteins database for the investigation of sequences and structures, *J. Mol. Biol.*, **247**(4), 536-540.

[18] Pollastri,G., Przybylski,D., Rost,B., Baldi,P. (2002) Improving the prediction of protein secondary structure in three and eight classes using recurrent neural networks and profiles, *Proteins*, **47**(2), 228-235.

[19] Pollastri,G., Baldi,P., Fariselli,P., Casadio,R. (2002) Prediction of Coordination Number and Relative Solvent Accessibility in Proteins, *Proteins*, **47**(2), 142-153.

[20] Porto,M., Bastolla,U., Roman,H.E., Vendruscolo,M. (2004) Reconstruction of protein structures from a vectorial representation, *Phys. Rev. Lett.*, **92**, 218101.

[21] Punta,M., Rost,B. (2005) PROFcon: novel prediction of long-range contacts, *Bioinformatics*, **21**, 2960-2968.)

[22] Ross,S., Munoz,D., Hebert,M., Bagnell,J.A. (2011) Learning message-passing inference machines for structured prediction, Proceedings of the 2011 IEEE Conference on Computer Vision and Pattern Recognition, 2737-2744.

[23] Sathyapriya,R., Duarte,J.M., Stehr,H., Filippis,I., Lappe,M. (2009) Defining an Essence of Structure Determining Residue Contacts in Proteins. *PLoS Comput Biol*, **5**(12), e1000584.

[24] Shackelford,G., Karplus, K. (2007) Contact prediction using mutual information and neural nets.*Proteins*, **69**,159-164.

[25] Tress,M.L., Valencia,A. (2010) Predicted residue-residue contacts can help the scoring of 3D models. *Proteins*, **78**(8), 1980-1991.

[26] Vassura,M., Margara,L., Di Lena,P., Medri,F., Fariselli,P. , Casadio,R. (2008) FT-COMAR: fault tolerant three-dimensional structure reconstruction from protein contact maps. *Bioinformatics*, **24**, 1313-1315.

[27] Zhang,Y. (2008) Progress and challenges in protein structure prediction. *Curr Opin Struct Biol.*, *18*(3), 342-348.

